# Networks with Learned Unit Response Functions

**John Moody and Norman Yarvin**
Yale Computer Science, 51 Prospect St.
P.O. Box 2158 Yale Station, New Haven, CT 06520-2158

## Abstract

Feedforward networks composed of units which compute a sigmoidal function of a weighted sum of their inputs have been much investigated. We tested the approximation and estimation capabilities of networks using functions more complex than sigmoids. Three classes of functions were tested: polynomials, rational functions, and flexible Fourier series. Unlike sigmoids, these classes can fit non-monotonic functions. They were compared on three problems: prediction of Boston housing prices, the sunspot count, and robot arm inverse dynamics. The complex units attained clearly superior performance on the robot arm problem, which is a highly non-monotonic, pure approximation problem. On the noisy and only mildly nonlinear Boston housing and sunspot problems, differences among the complex units were revealed; polynomials did poorly, whereas rationals and flexible Fourier series were comparable to sigmoids.

## 1  Introduction

A commonly studied neural architecture is the feedforward network in which each unit of the network computes a nonlinear function $g(x)$ of a weighted sum of its inputs $x = w^t u$. Generally this function is a sigmoid, such as $g(x) = \tanh x$ or $g(x) = 1/(1 + e^{(x-\theta)})$. To these we compared units of a substantially different type: they also compute a nonlinear function of a weighted sum of their inputs, but the unit response function is able to fit a much higher degree of nonlinearity than can a sigmoid. The nonlinearities we considered were polynomials, rational functions (ratios of polynomials), and flexible Fourier series (sums of cosines.) Our comparisons were done in the context of two-layer networks consisting of one hidden layer of complex units and an output layer of a single linear unit.

This network architecture is similar to that built by projection pursuit regression (PPR) [1, 2], another technique for function approximation. The one difference is that in PPR the nonlinear function of the units of the hidden layer is a nonparametric smooth. This nonparametric smooth has two disadvantages for neural modeling: it has many parameters, and, as a smooth, it is easily trained only if desired output values are available for that particular unit. The latter property makes the use of smooths in multilayer networks inconvenient. If a parametrized function of a type suitable for one-dimensional function approximation is used instead of the nonparametric smooth, then these disadvantages do not apply. The functions we used are all suitable for one-dimensional function approximation.

## 2 Representation

A few details of the representation of the unit response functions are worth noting.

**Polynomials:** Each polynomial unit computed the function

$$g(x) = a_1 x + a_2 x^2 + ... + a_n x^n$$

with $x = w^T u$ being the weighted sum of the input. A zero'th order term was not included in the above formula, since it would have been redundant among all the units. The zero'th order term was dealt with separately and only stored in one location.

**Rationals:** A rational function representation was adopted which could not have zeros in the denominator. This representation used a sum of squares of polynomials, as follows:

$$g(x) = \frac{a_0 + a_1 x + ... + a_n x^n}{1 + (b_0 + b_1 x)^2 + (b_2 x + b_3 x^2)^2 + (b_4 x + b_5 x^2 + b_6 x^3 + b_7 x^4)^2 + ...}$$

This representation has the qualities that the denominator is never less than 1, and that $n$ parameters are used to produce a denominator of degree $n$. If the above formula were continued the next terms in the denominator would be of degrees eight, sixteen, and thirty-two. This powers-of-two sequence was used for the following reason: of the $2(n - m)$ terms in the square of a polynomial $p = a_m x^m + ... + a_n x^n$, it is possible by manipulating $a_m...a_n$ to determine the $n - m$ highest coefficients, with the exception that the very highest coefficient must be non-negative. Thus if we consider the coefficients of the polynomial that results from squaring and adding together the terms of the denominator of the above formula, the highest degree squared polynomial may be regarded as determining the highest half of the coefficients, the second highest degree polynomial may be regarded as determining the highest half of the rest of the coefficients, and so forth. This process cannot set all the coefficients arbitrarily; some must be non-negative.

**Flexible Fourier series:** The flexible Fourier series units computed

$$g(x) = \sum_{i=0}^{n} a_i \cos(b_i x + c_i)$$

where the amplitudes $a_i$, frequencies $b_i$ and phases $c_i$ were unconstrained and could assume any value.

**Sigmoids:** We used the standard logistic function:

$$g(x) = 1/(1 + e^{(x-\theta)})$$

# 3    Training Method

All the results presented here were trained with the Levenberg-Marquardt modification of the Gauss-Newton nonlinear least squares algorithm. Stochastic gradient descent was also tried at first, but on the problems where the two were compared, Levenberg-Marquardt was much superior both in convergence time and in quality of result. Levenberg-Marquardt required substantially fewer iterations than stochastic gradient descent to converge. However, it needs $O(p^2)$ space and $O(p^2n)$ time per iteration in a network with $p$ parameters and $n$ input examples, as compared to $O(p)$ space and $O(pn)$ time per epoch for stochastic gradient descent. Further details of the training method will be discussed in a longer paper.

With some data sets, a weight decay term was added to the energy function to be optimized. The added term was of the form $\lambda \sum_{i=1}^{n} w_i^2$. When weight decay was used, a range of values of $\lambda$ was tried for every network trained.

Before training, all the data was normalized: each input variable was scaled so that its range was (-1,1), then scaled so that the maximum sum of squares of input variables for any example was 1. The output variable was scaled to have mean zero and mean absolute value 1. This helped the training algorithm, especially in the case of stochastic gradient descent.

# 4    Results

We present results of training our networks on three data sets: robot arm inverse dynamics, Boston housing data, and sunspot count prediction. The Boston and sunspot data sets are noisy, but have only mild nonlinearity. The robot arm inverse dynamics data has no noise, but a high degree of nonlinearity. Noise-free problems have low estimation error. Models for linear or mildly nonlinear problems typically have low approximation error. The robot arm inverse dynamics problem is thus a pure approximation problem, while performance on the noisy Boston and sunspots problems is limited more by estimation error than by approximation error.

Figure 1a is a graph, as those used in PPR, of the unit response function of a one-unit network trained on the Boston housing data. The x axis is a projection (a weighted sum of inputs $w^T u$) of the 13-dimensional input space onto 1 dimension, using those weights chosen by the unit in training. The y axis is the fit to data. The response function of the unit is a sum of three cosines. Figure 1b is the superposition of five graphs of the five unit response functions used in a five-unit rational function solution (RMS error less than 2%) of the robot arm inverse dynamics problem. The domain for each curve lies along a different direction in the six-dimensional input space. Four of the five fits along the projection directions are non-monotonic, and thus can be fit only poorly by a sigmoid.

Two different error measures are used in the following. The first is the RMS error, normalized so that error of 1 corresponds to no training. The second measure is the

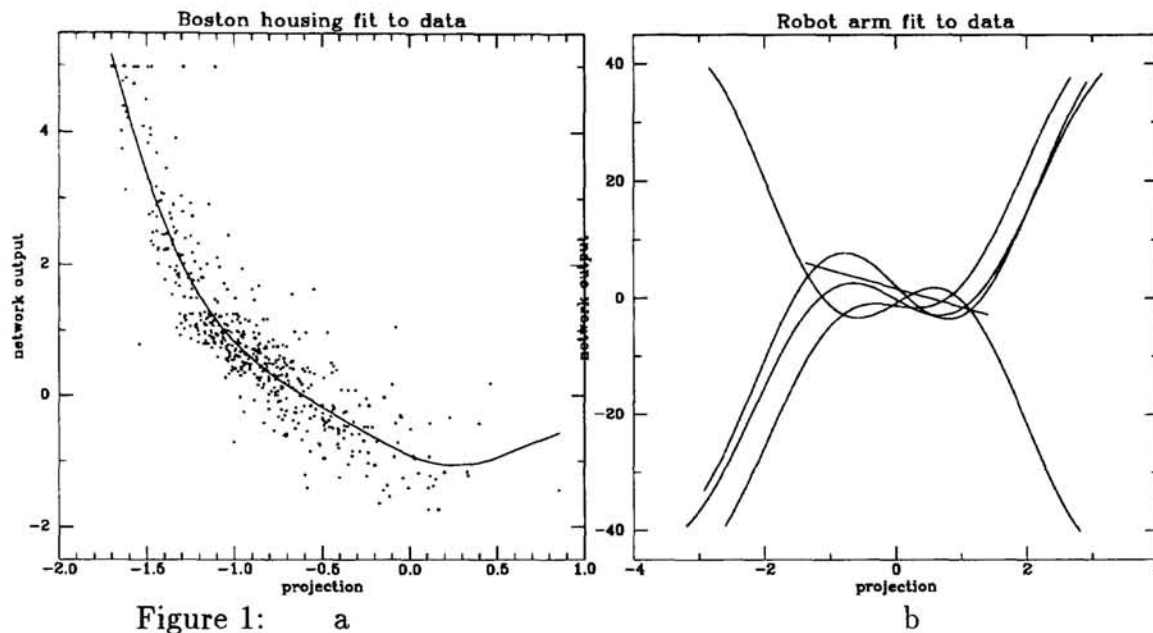

Figure 1:    a                                      b

square of the normalized RMS error, otherwise known as the fraction of explained variance. We used whichever error measure was used in earlier work on that data set.

## 4.1   Robot arm inverse dynamics

This problem is the determination of the torque necessary at the joints of a two-joint robot arm required to achieve a given acceleration of each segment of the arm, given each segment's velocity and position. There are six input variables to the network, and two output variables. This problem was treated as two separate estimation problems, one for the shoulder torque and one for the elbow torque. The shoulder torque was a slightly more difficult problem, for almost all networks. The 1000 points in the training set covered the input space relatively thoroughly. This, together with the fact that the problem had no noise, meant that there was little difference between training set error and test set error.

Polynomial networks of limited degree are not universal approximators, and that is quite evident on this data set; polynomial networks of low degree reached their minimum error after a few units. Figure 2a shows this. If polynomial, cosine, rational, and sigmoid networks are compared as in Figure 2b, leaving out low degree polynomials, the sigmoids have relatively high approximation error even for networks with 20 units. As shown in the following table, the complex units have more parameters each, but still get better performance with fewer parameters total.

| Type | Units | Parameters | Error |
|---|---|---|---|
| degree 7 polynomial | 5 | 65 | .024 |
| degree 6 rational | 5 | 95 | .027 |
| 2 term cosine | 6 | 73 | .020 |
| sigmoid | 10 | 81 | .139 |
| sigmoid | 20 | 161 | .119 |

Since the training set is noise-free, these errors represent pure approximation error.

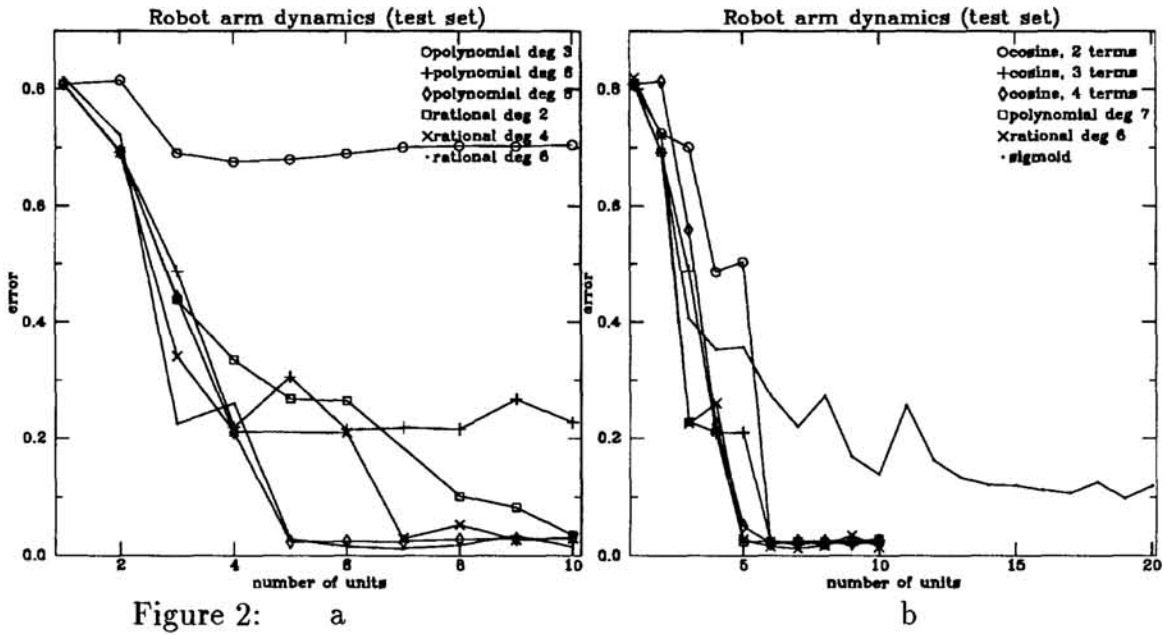

Figure 2:     a                                                    b

The superior performance of the complex units on this problem is probably due to their ability to approximate non-monotonic functions.

## 4.2   Boston housing

The second data set is a benchmark for statistical algorithms: the prediction of Boston housing prices from 13 factors [3]. This data set contains 506 exemplars and is relatively simple; it can be approximated well with only a single unit. Networks of between one and six units were trained on this problem. Figure 3a is a graph of training set performance from networks trained on the entire data set; the error measure used was the fraction of explained variance. From this graph it is apparent

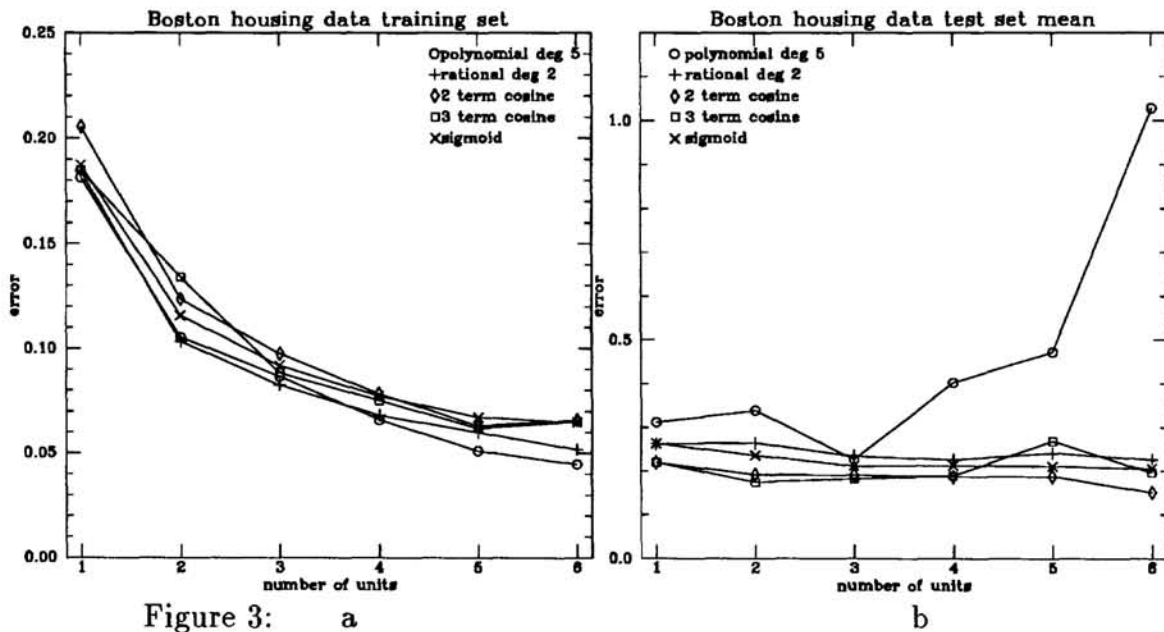

Figure 3:     a                                                    b

that training set performance does not vary greatly between different types of units, though networks with more units do better.

On the test set there is a large difference. This is shown in Figure 3b. Each point on the graph is the average performance of ten networks of that type. Each network was trained using a different permutation of the data into test and training sets, the test set being 1/3 of the examples and the training set 2/3. It can be seen that the cosine nets perform the best, the sigmoid nets a close second, the rationals third, and the polynomials worst (with the error increasing quite a bit with increasing polynomial degree.)

It should be noted that the distribution of errors is far from a normal distribution, and that the training set error gives little clue as to the test set error. The following table of errors, for nine networks of four units using a degree 5 polynomial, is somewhat typical:

| Set | Error | | | | | | | | |
|-----|-------|-------|-------|-------|-------|-------|-------|-------|-------|
| training | 0.095 | 0.062 | 0.060 | 0.090 | 0.076 | 0.065 | 0.068 | 0.066 | 0.091 |
| test | 0.085 | 1.677 | 0.171 | 0.197 | 0.143 | 0.546 | 0.250 | 0.158 | 0.395 |

Our speculation on the cause of these extremely high errors is that polynomial approximations do not extrapolate well; if the prediction of some data point results in a polynomial being evaluated slightly outside the region on which the polynomial was trained, the error may be extremely high. Rational functions where the numerator and denominator have equal degree have less of a problem with this, since asymptotically they are constant. However, over small intervals they can have the extrapolation characteristics of polynomials. Cosines are bounded, and so, though they may not extrapolate well if the function is not somewhat periodic, at least do not reach large values like polynomials.

## 4.3  Sunspots

The third problem was the prediction of the average monthly sunspot count in a given year from the values of the previous twelve years. We followed previous work in using as our error measure the fraction of variance explained, and in using as the training set the years 1700 through 1920 and as the test set the years 1921 through 1955. This was a relatively easy test set – every network of one unit which we trained (whether sigmoid, polynomial, rational, or cosine) had, in each of ten runs, a training set error between .147 and .153 and a test set error between .105 and .111. For comparison, the best test set error achieved by us or previous testers was about .085. A similar set of runs was done as those for the Boston housing data, but using at most four units; similar results were obtained. Figure 4a shows training set error and Figure 4b shows test set error on this problem.

## 4.4  Weight Decay

The performance of almost all networks was improved by some amount of weight decay. Figure 5 contains graphs of test set error for sigmoidal and polynomial units,

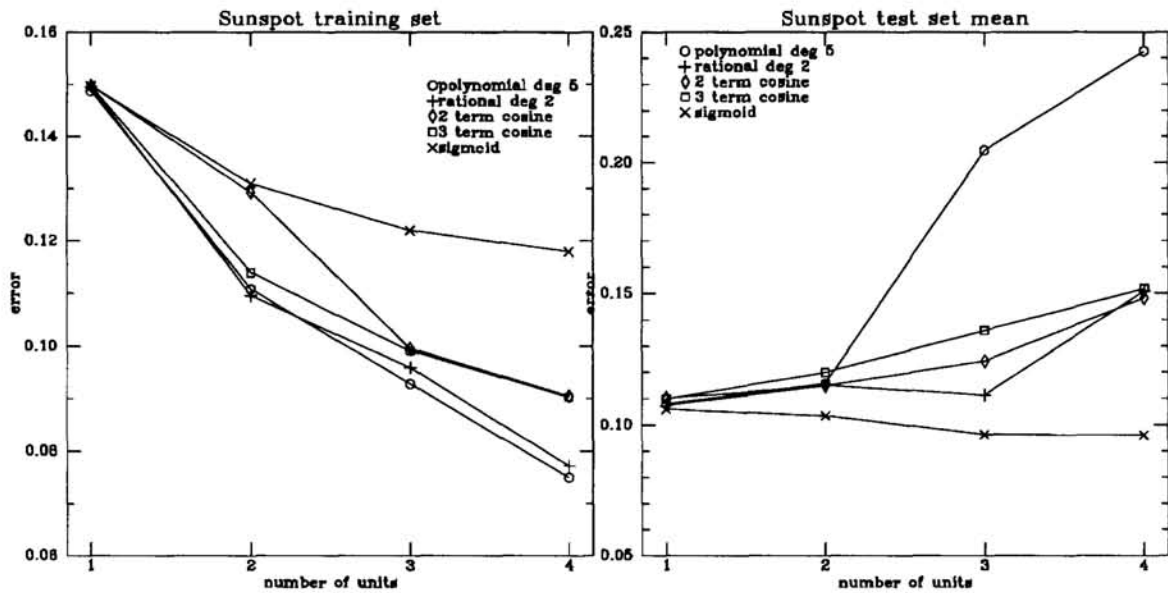

Figure 4:     a                                                b

using various values of the weight decay parameter $\lambda$. For the sigmoids, very little weight decay seems to be needed to give good results, and there is an order of magnitude range (between .001 and .01) which produces close to optimal results. For polynomials of degree 5, more weight decay seems to be necessary for good results; in fact, the highest value of weight decay is the best. Since very high values of weight decay are needed, and at those values there is little improvement over using a single unit, it may be supposed that using those values of weight decay restricts the multiple units to producing a very similar solution to the one-unit solution. Figure 6 contains the corresponding graphs for sunspots. Weight decay seems to help less here for the sigmoids, but for the polynomials, moderate amounts of weight decay produce an improvement over the one-unit solution.

## Acknowledgements

The authors would like to acknowledge support from ONR grant N00014-89-J-1228, AFOSR grant 89-0478, and a fellowship from the John and Fannie Hertz Foundation. The robot arm data set was provided by Chris Atkeson.

## References

[1] J. H. Friedman, W. Stuetzle, "Projection Pursuit Regression", *Journal of the American Statistical Association*, December 1981, Volume 76, Number 376, 817-823

[2] P. J. Huber, "Projection Pursuit", *The Annals of Statistics*, 1985 Vol. 13 No. 2, 435-475

[3] L. Breiman et al, *Classification and Regression Trees*, Wadsworth and Brooks, 1984, pp217-220

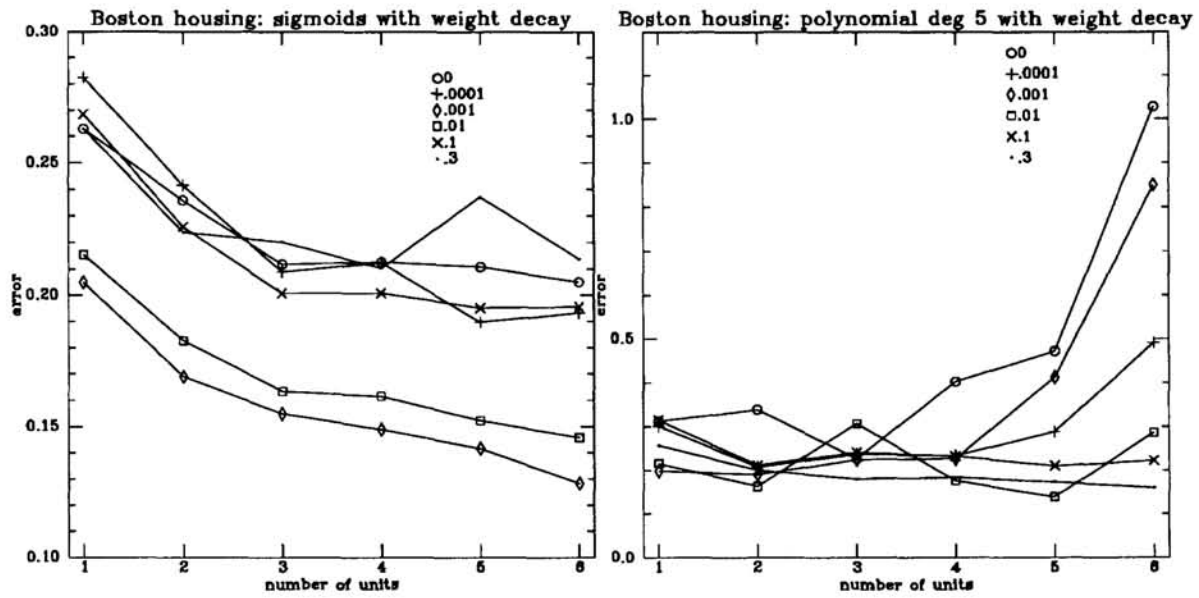

Figure 5: Boston housing test error with various amounts of weight decay

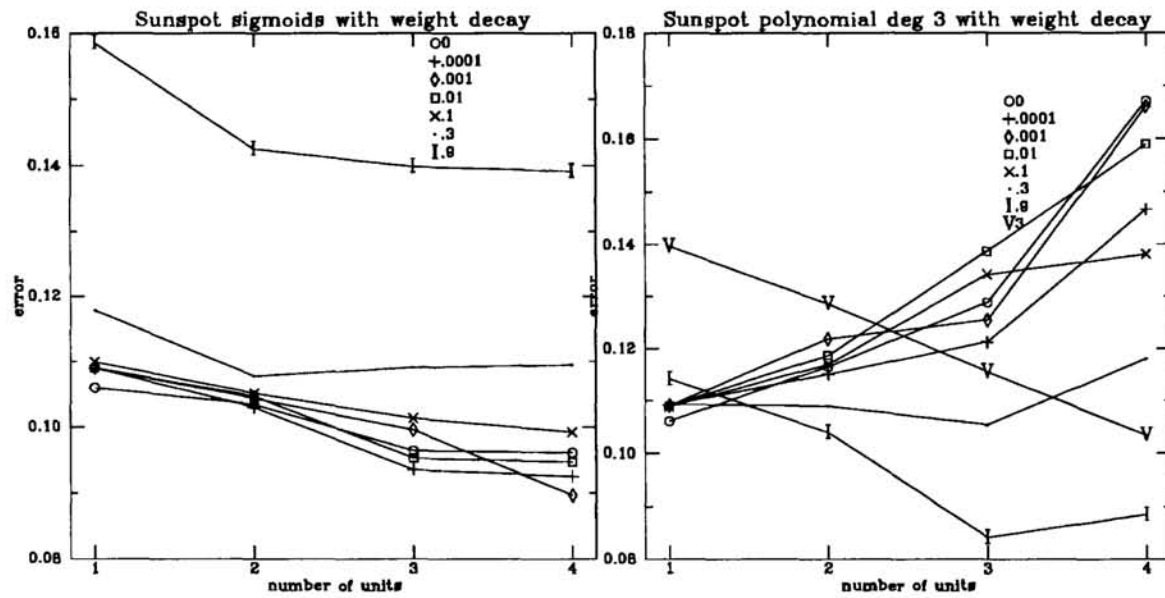

Figure 6: Sunspot test error with various amounts of weight decay